# Emergence of Topography and Complex Cell Properties from Natural Images using Extensions of ICA

**Aapo Hyvärinen and Patrik Hoyer**
Neural Networks Research Center
Helsinki University of Technology
P.O. Box 5400, FIN-02015 HUT, Finland
`aapo.hyvarinen@hut.fi, patrik.hoyer@hut.fi`
`http://www.cis.hut.fi/projects/ica/`

## Abstract

Independent component analysis of natural images leads to emergence of simple cell properties, i.e. linear filters that resemble wavelets or Gabor functions. In this paper, we extend ICA to explain further properties of V1 cells. First, we decompose natural images into independent subspaces instead of scalar components. This model leads to emergence of phase and shift invariant features, similar to those in V1 complex cells. Second, we define a topography between the linear components obtained by ICA. The topographic distance between two components is defined by their higher-order correlations, so that two components are close to each other in the topography if they are strongly dependent on each other. This leads to simultaneous emergence of both topography and invariances similar to complex cell properties.

## 1 Introduction

A fundamental approach in signal processing is to design a statistical generative model of the observed signals. Such an approach is also useful for modeling the properties of neurons in primary sensory areas. The basic models that we consider here express a static monochrome image $I(x,y)$ as a linear superposition of some features or basis functions $b_i(x,y)$:

$$I(x,y) = \sum_{i=1}^{n} b_i(x,y)s_i \qquad (1)$$

where the $s_i$ are stochastic coefficients, different for each image $I(x,y)$. Estimation of the model in Eq. (1) consists of determining the values of $s_i$ and $b_i(x,y)$ for all $i$ and $(x,y)$, given a sufficient number of observations of images, or in practice, image patches $I(x,y)$. We restrict ourselves here to the basic case where the $b_i(x,y)$ form an invertible linear system. Then we can invert $s_i = <w_i, I>$ where the $w_i$ denote the inverse filters, and $<w_i, I> = \sum_{x,y} w_i(x,y)I(x,y)$ denotes the dot-product.

The $w_i(x, y)$ can then be identified as the receptive fields of the model simple cells, and the $s_i$ are their activities when presented with a given image patch $I(x, y)$.

In the basic case, we assume that the $s_i$ are nongaussian, and mutually independent. This type of decomposition is called independent component analysis (ICA) [3, 9, 1, 8], or sparse coding [13]. Olshausen and Field [13] showed that when this model is estimated with input data consisting of patches of natural scenes, the obtained filters $w_i(x, y)$ have the three principal properties of simple cells in V1: they are localized, oriented, and bandpass (selective to scale/frequency). Van Hateren and van der Schaaf [15] compared quantitatively the obtained filters $w_i(x, y)$ with those measured by single-cell recordings of the macaque cortex, and found a good match for most of the parameters.

We show in this paper that simple extensions of the basic ICA model explain emergence of further properties of V1 cells: topography and the invariances of complex cells. Due to space limitations, we can only give the basic ideas in this paper. More details can be found in [6, 5, 7].

First, using the method of feature subspaces [11], we model the response of a complex cell as the norm of the projection of the input vector (image patch) onto a linear subspace, which is equivalent to the classical energy models. Then we maximize the independence between the norms of such projections, or energies. Thus we obtain features that are localized in space, oriented, and bandpass, like those given by simple cells, or Gabor analysis. In contrast to simple linear filters, however, the obtained feature subspaces also show emergence of phase invariance and (limited) shift or translation invariance. Maximizing the independence, or equivalently, the sparseness of the norms of the projections to feature subspaces thus allows for the emergence of exactly those invariances that are encountered in complex cells.

Second, we extend this model of independent subspaces so that we have overlapping subspaces, and every subspace corresponds to a neighborhood on a topographic grid. This is called topographic ICA, since it defines a topographic organization between components. Components that are far from each other on the grid are independent, like in ICA. In contrast, components that are near to each other are not independent: they have strong higher-order correlations. This model shows emergence of both complex cell properties and topography from image data.

## 2   Independent subspaces as complex cells

In addition to the simple cells that can be modelled by basic ICA, another important class of cells in V1 is complex cells. The two principal properties that distinguish complex cells from simple cells are phase invariance and (limited) shift invariance. The purpose of the first model in this paper is to explain the emergence of such phase and shift invariant features using a modification of the ICA model. The modification is based on combining the principle of invariant-feature subspaces [11] and the model of multidimensional independent component analysis [2].

**Invariant feature subspaces.** The principle of invariant-feature subspaces states that one may consider an invariant feature as a linear subspace in a feature space. The value of the invariant, higher-order feature is given by (the square of) the norm of the projection of the given data point on that subspace, which is typically spanned by lower-order features. A feature subspace, as any linear subspace, can always be represented by a set of orthogonal basis vectors, say $w_i(x, y), i = 1, ..., m$, where $m$ is the dimension of the subspace. Then the value $F(I)$ of the feature $F$ with input vector $I(x, y)$ is given by $F(I) = \sum_{i=1}^{m} < w_i, I >^2$, where a square root

might be taken. In fact, this is equivalent to computing the distance between the input vector $I(x,y)$ and a general linear combination of the basis vectors (filters) $w_i(x,y)$ of the feature subspace [11]. In [11], it was shown that this principle, when combined with competitive learning techniques, can lead to emergence of invariant image features.

**Multidimensional independent component analysis.** In multidimensional independent component analysis [2] (see also [12]), a linear generative model as in Eq. (1) is assumed. In contrast to ordinary ICA, however, the components (responses) $s_i$ are not assumed to be all mutually independent. Instead, it is assumed that the $s_i$ can be divided into couples, triplets or in general $m$-tuples, such that the $s_i$ inside a given $m$-tuple may be dependent on each other, but dependencies between different $m$-tuples are not allowed. Every $m$-tuple of $s_i$ corresponds to $m$ basis vectors $b_i(x,y)$. The $m$-dimensional probability densities inside the $m$-tuples of $s_i$ is not specified in advance in the general definition of multidimensional ICA [2]. In the following, let us denote by $J$ the number of independent feature subspaces, and by $S_j, j = 1, ..., J$ the set of the indices of the $s_i$ belonging to the subspace of index $j$.

**Independent feature subspaces.** Invariant-feature subspaces can be embedded in multidimensional independent component analysis by considering probability distributions for the $m$-tuples of $s_i$ that are *spherically symmetric*, i.e. depend only on the norm. In other words, the probability density $p_j(.)$ of the $m$-tuple with index $j \in \{1, ..., J\}$, can be expressed as a function of the sum of the squares of the $s_i, i \in S_j$ only. For simplicity, we assume further that the $p_j(.)$ are equal for all $j$, i.e. for all subspaces.

Assume that the data consists of $K$ observed image patches $I_k(x,y), k = 1, ..., K$. Then the logarithm of the likelihood $L$ of the data given the model can be expressed as

$$\log L(w_i(x,y), i = 1...n) = \sum_{k=1}^{K}\sum_{j=1}^{J} \log p(\sum_{i \in S_j} <w_i, I_k>^2) + K \log |\det \mathbf{W}| \qquad (2)$$

where $p(\sum_{i \in S_j} s_i^2) = p_j(s_i, i \in S_j)$ gives the probability density inside the $j$-th $m$-tuple of $s_i$, and $\mathbf{W}$ is a matrix containing the filters $w_i(x,y)$ as its columns.

As in basic ICA, prewhitening of the data allows us to consider the $w_i(x,y)$ to be orthonormal, and this implies that $\log |\det \mathbf{W}|$ is zero [6]. Thus we see that the likelihood in Eq. (2) is a function of the norms of the projections of $I_k(x,y)$ on the subspaces indexed by $j$, which are spanned by the orthonormal basis sets given by $w_i(x,y), i \in S_j$. Since the norm of the projection of visual data on practically any subspace has a supergaussian distribution, we need to choose the probability density $p$ in the model to be sparse [13], i.e. supergaussian [8]. For example, we could use the following probability distribution

$$\log p(\sum_{i \in S_j} s_i^2) = -\alpha[\sum_{i \in S_j} s_i^2]^{1/2} + \beta, \qquad (3)$$

which could be considered a multi-dimensional version of the exponential distribution. Now we see that the estimation of the model consists of finding subspaces such that the *norms of the projections of the (whitened) data on those subspaces have maximally sparse distributions*.

The introduced "independent (feature) subspace analysis" is a natural generalization of ordinary ICA. In fact, if the projections on the subspaces are reduced to dot-products, i.e. projections on 1-D subspaces, the model reduces to ordinary ICA

(provided that, in addition, the independent components are assumed to have non-skewed distributions). It is to be expected that the norms of the projections on the subspaces represent some higher-order, invariant features. The exact nature of the invariances has not been specified in the model but will emerge from the input data, using only the prior information on their independence.

When independent subspace analysis is applied to natural image data, we can identify the norms of the projections $(\sum_{i \in S_j} s_i^2)^{1/2}$ as the responses of the complex cells. If the individual filter vectors $w_i(x, y)$ are identified with the receptive fields of simple cells, this can be interpreted as a hierarchical model where the complex cell response is computed from simple cell responses $s_i$, in a manner similar to the classical energy models for complex cells. Experiments (see below and [6]) show that the model does lead to emergence of those invariances that are encountered in complex cells.

## 3   Topographic ICA

The independent subspace analysis model introduces a certain dependence structure for the components $s_i$. Let us assume that the distribution in the subspace is sparse, which means that the norm of the projection is most of the time very near to zero. This is the case, for example, if the densities inside the subspaces are specified as in (3). Then the model implies that two components $s_i$ and $s_j$ that belong to the same subspace tend to be nonzero simultaneously. In other words, $s_i^2$ and $s_j^2$ are positively correlated. This seems to be a preponderant structure of dependency in most natural data. For image data, this has also been noted by Simoncelli [14].

Now we generalize the model defined by (2) so that it models this kind of dependence not only inside the $m$-tuples, but among all "neighboring" components. A neighborhood relation defines a topographic order [10]. (A different generalization based on an explicit generative model is given in [5].) We define the model by the following likelihood:

$$\log L(w_i(x, y), i = 1, ..., n) = \sum_{k=1}^{K} \sum_{j=1}^{n} G(\sum_{i=1}^{n} h(i, j) < w_i, I_k >^2) + K \log|\det \mathbf{W}| \quad (4)$$

Here, $h(i, j)$ is a neighborhood function, which expresses the strength of the connection between the $i$-th and $j$-th units. The neighborhood function can be defined in the same way as with the self-organizing map [10]. Neighborhoods can thus be defined as one-dimensional or two-dimensional; 2-D neighborhoods can be square or hexagonal. A simple example is to define a 1-D neighborhood relation by

$$h(i, j) = \begin{cases} 1, & \text{if } |i - j| \leq m \\ 0, & \text{otherwise.} \end{cases} \quad (5)$$

The constant $m$ defines here the width of the neighborhood.

The function $G$ has a similar role as the log-density of the independent components in classic ICA. For image data, or other data with a sparse structure, $G$ should be chosen as in independent subspace analysis, see Eq. (3).

**Properties of the topographic ICA model.** Here, we consider for simplicity only the case of sparse data. The first basic property is that all the components $s_i$ are uncorrelated, as can be easily proven by symmetry arguments [5]. Moreover, their variances can be defined to be equal to unity, as in classic ICA. Second, components $s_i$ and $s_j$ that are near to each other, i.e. such that $h(i, j)$ is significantly non-zero,

tend to be active (non-zero) at the same time. In other words, their energies $s_i^2$ and $s_j^2$ are positively correlated. Third, latent variables that are far from each other are practically independent. Higher-order correlation decreases as a function of distance, assuming that the neighborhood is defined in a way similar to that in (5). For details, see [5].

Let us note that our definition of *topography by higher-order correlations* is very different from the one used in practically all existing topographic mapping methods. Usually, the distance is defined by basic geometrical relations like Euclidean distance or correlation. Interestingly, our principle makes it possible to define a topography even among a set of orthogonal vectors whose Euclidean distances are all equal. Such orthogonal vectors are actually encountered in ICA, where the basis vectors and filters can be constrained to be orthogonal in the whitened space.

# 4 Experiments with natural image data

We applied our methods on natural image data. The data was obtained by taking $16 \times 16$ pixel image patches at random locations from monochrome photographs depicting wild-life scenes (animals, meadows, forests, etc.). Preprocessing consisted of removing the DC component and reducing the dimension of the data to 160 by PCA. For details on the experiments, see [6, 5].

Fig. 1 shows the basis vectors of the 40 feature subspaces (complex cells), when subspace dimension was chosen to be 4. It can be seen that the basis vectors associated with a single complex cell all have approximately the same orientation and frequency. Their locations are not identical, but close to each other. The phases differ considerably. Every feature subspace can thus be considered a generalization of a quadrature-phase filter pair as found in the classical energy models, enabling the cell to be selective to some given orientation and frequency, but invariant to phase and somewhat invariant to shifts. Using 4 dimensions instead of 2 greatly enhances the shift invariance of the feature subspace.

In topographic ICA, the neighborhood function was defined so that every neighborhood consisted of a $3 \times 3$ square of 9 units on a 2-D torus lattice [10]. The obtained basis vectors, are shown in Fig. 2. The basis vectors are similar to those obtained by ordinary ICA of image data [13, 1]. In addition, they have a clear topographic organization. In addition, the connection to independent subspace analysis is clear from Fig. 2. Two neighboring basis vectors in Fig. 2 tend to be of the same orientation and frequency. Their locations are near to each other as well. In contrast, their phases are very different. This means that a neighborhood of such basis vectors, i.e. simple cells, is similar to an independent subspace. Thus it functions as a complex cell. This was demonstrated in detail in [5].

# 5 Discussion

We introduced here two extensions of ICA that are especially useful for image modelling. The first model uses a subspace representation to model invariant features. It turns out that the independent subspaces of natural images are similar to complex cells. The second model is a further extension of the independent subspace model. This topographic ICA model is a generative model that combines topographic mapping with ICA. As in all topographic mappings, the distance in the representation space (on the topographic "grid") is related to some measure of distance between represented components. In topographic ICA, the distance between represented components is defined by higher-order correlations, which gives

the natural distance measure in the context of ICA.

An approach closely related to ours is given by Kohonen's Adaptive Subspace Self-Organizing Map [11]. However, the emergence of shift invariance in [11] was conditional to restricting consecutive patches to come from nearby locations in the image, giving the input data a temporal structure like in a smoothly changing image sequence. Similar developments were given by Földiák [4]. In contrast to these two theories, we formulated an explicit image model. This independent subspace analysis model shows that emergence of complex cell properties is possible using patches at random, independently selected locations, which proves that there is enough information in static images to explain the properties of complex cells. Moreover, by extending this subspace model to model topography, we showed that *the emergence of both topography and complex cell properties can be explained by a single principle*: neighboring cells should have strong higher-order correlations.

# References

[1] A.J. Bell and T.J. Sejnowski. The 'independent components' of natural scenes are edge filters. *Vision Research*, 37:3327–3338, 1997.

[2] J.-F. Cardoso. Multidimensional independent component analysis. In *Proc. IEEE Int. Conf. on Acoustics, Speech and Signal Processing (ICASSP'98)*, Seattle, WA, 1998.

[3] P. Comon. Independent component analysis – a new concept? *Signal Processing*, 36:287–314, 1994.

[4] P. Földiák. Learning invariance from transformation sequences. *Neural Computation*, 3:194–200, 1991.

[5] A. Hyvärinen and P. O. Hoyer. Topographic independent component analysis. 1999. Submitted, available at http://www.cis.hut.fi/~aapo/.

[6] A. Hyvärinen and P. O. Hoyer. Emergence of phase and shift invariant features by decomposition of natural images into independent feature subspaces. *Neural Computation*, 2000. (in press).

[7] A. Hyvärinen, P. O. Hoyer, and M. Inki. The independence assumption: Analyzing the independence of the components by topography. In M. Girolami, editor, *Advances in Independent Component Analysis*. Springer-Verlag, 2000. in press.

[8] A. Hyvärinen and E. Oja. A fast fixed-point algorithm for independent component analysis. *Neural Computation*, 9(7):1483–1492, 1997.

[9] C. Jutten and J. Herault. Blind separation of sources, part I: An adaptive algorithm based on neuromimetic architecture. *Signal Processing*, 24:1–10, 1991.

[10] T. Kohonen. *Self-Organizing Maps*. Springer-Verlag, Berlin, Heidelberg, New York, 1995.

[11] T. Kohonen. Emergence of invariant-feature detectors in the adaptive-subspace self-organizing map. *Biological Cybernetics*, 75:281–291, 1996.

[12] J. K. Lin. Factorizing multivariate function classes. In *Advances in Neural Information Processing Systems*, volume 10, pages 563–569. The MIT Press, 1998.

[13] B. A. Olshausen and D. J. Field. Emergence of simple-cell receptive field properties by learning a sparse code for natural images. *Nature*, 381:607–609, 1996.

[14] E. P. Simoncelli and O. Schwartz. Modeling surround suppression in V1 neurons with a statistically-derived normalization model. In *Advances in Neural Information Processing Systems 11*, pages 153–159. MIT Press, 1999.

[15] J. H. van Hateren and A. van der Schaaf. Independent component filters of natural images compared with simple cells in primary visual cortex. *Proc. Royal Society ser. B*, 265:359–366, 1998.

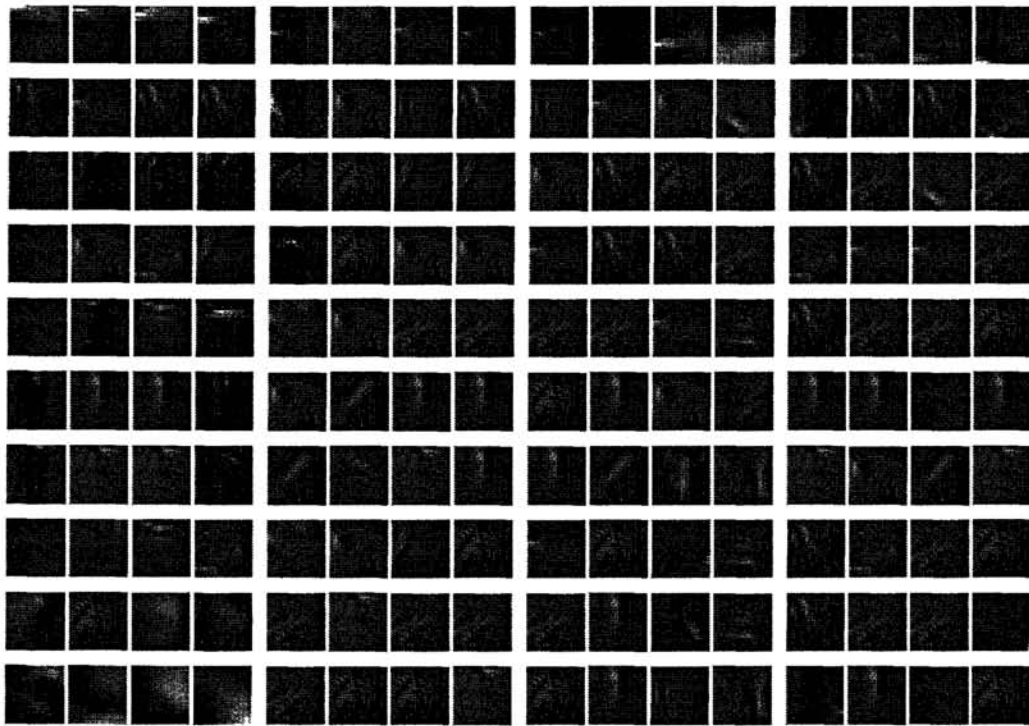

Figure 1: Independent subspaces of natural image data. The model gives Gabor-like basis vectors for image windows. Every group of four basis vectors corresponds to one independent feature subspace, or complex cell. Basis vectors in a subspace are similar in orientation, location and frequency. In contrast, their phases are very different.

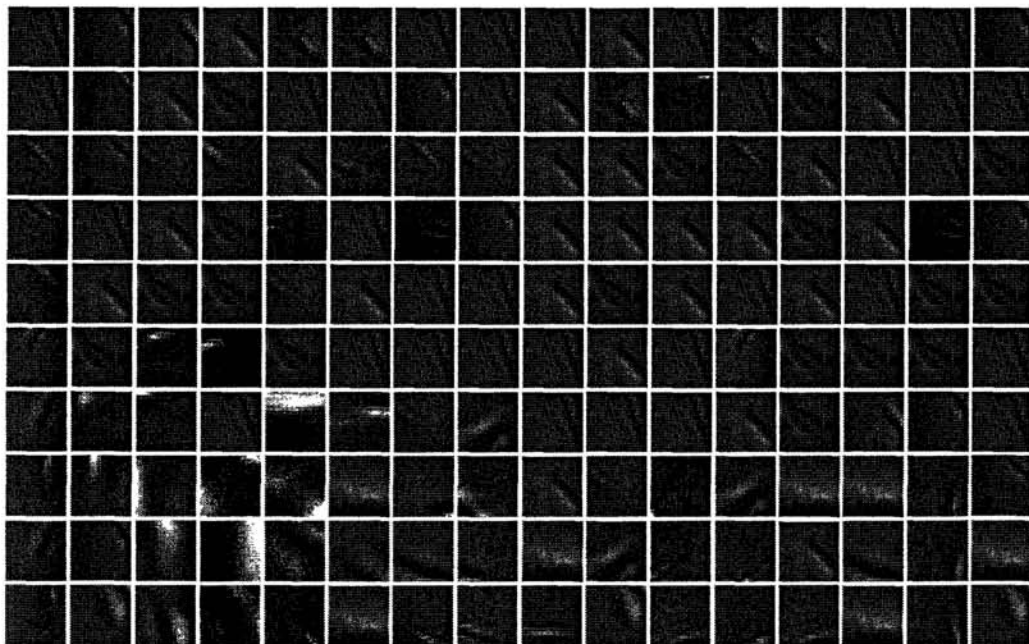

Figure 2: Topographic ICA of natural image data. This gives Gabor-like basis vectors as well. Basis vectors that are similar in orientation, location and/or frequency are close to each other. The phases of nearby basis vectors are very different, giving each neighborhood properties similar to a complex cell.